# Constructing Proofs in Symmetric Networks

Gadi Pinkas
Computer Science Department
Washington University
Campus Box 1045
St. Louis, MO 63130

## Abstract

This paper considers the problem of expressing predicate calculus in connectionist networks that are based on energy minimization. Given a first-order-logic knowledge base and a bound $k$, a symmetric network is constructed (like a Boltzman machine or a Hopfield network) that searches for a proof for a given query. If a resolution-based proof of length no longer than $k$ exists, then the global minima of the energy function that is associated with the network represent such proofs. The network that is generated is of size cubic in the bound $k$ and linear in the knowledge size. There are no restrictions on the type of logic formulas that can be represented. The network is inherently fault tolerant and can cope with inconsistency and nonmonotonicity.

## 1 Introduction

The ability to reason from acquired knowledge is undoubtedly one of the basic and most important components of human intelligence. Among the major tools for reasoning in the area of AI are deductive proof techniques. However, traditional methods are plagued by intractability, inability to learn and adjust, as well as by inability to cope with noise and inconsistency. A connectionist approach may be the missing link: fine grain, massively parallel architecture may give us real-time approximation; networks are potentially trainable and adjustable; and they may be made tolerant to noise as a result of their collective computation.

Most connectionist reasoning systems that implement parts of first-order logic (see for examples: [Hölldobler 90], [Shastri et al. 90]) use the spreading activation paradigm and usually trade expressiveness with time efficiency. In contrast, this paper uses the energy minimization paradigm (like [Derthick 88], [Ballard 86] and [Pinkas 91c]), representing an intractable problem, but trading time with correctness; i.e., as more time is given, the probability of converging to a correct answer increases.

Symmetric connectionist networks used for constraint satisfaction are the target platform [Hopfield 84b], [Hinton, Sejnowski 86], [Peterson, Hartman 89], [Smolensky 86]. They are characterized by a quadratic energy function that should be minimized. Some of the models in the family may be seen as performing a search for a *global* minimum of their energy function. The task is therefore to represent logic deduction that is bound by a *finite* proof length as energy minimization (without a bound on the proof length, the problem is undecidable). When a query is clamped, the network should search for a proof that supports the query. If a proof to the query exists, then every global minimum of the energy function associated with the network represents a proof. If no proof exists, the global minima represent the lack of a proof.

The paper elaborates the propositional case; however, due to space limitations, the first-order (FOL) case is only sketched. For more details and full treatment of FOL see [Pinkas 91j].

## 2   Representing proofs of propositional logic

I'll start by assuming that the knowledge base is propositional.

**The proof area:**
A proof is a list of clauses ending with the query such that every clause used is either an original clause, a copy (or weakening) of a clause that appears earlier in the proof, or a result of a resolution step of the two clauses that appeared just earlier. The proof emerges as an activation pattern on special unit structures called the proof area, and is represented in reverse to the common practice (the query appears first). For example: given a knowledge base of the following clauses:
1) $A$
2) $\neg A \vee B \vee C$
3) $\neg B \vee D$
4) $\neg C \vee D$
we would like to prove the query $D$, by generating the following list of clauses:

1) $D$                 (obtained by resolution of clauses 2 and 3 by canceling $A$).
2) $A$                 (original clause no. 1).
3) $\neg A \vee D$          (obtained by resolution of clauses 4 and 5 by canceling $C$).
4) $\neg C \vee D$          (original clause no. 4).
5) $\neg A \vee C \vee D$       (obtained by resolution of clauses 6 and 7 by canceling $B$).
6) $\neg B \vee D$          (original clause no. 3).
7) $\neg A \vee B \vee C$       (original clause no. 2).

Each clause in the proof is either an original clause, a copy of a clause from earlier in the proof, or a resolution step.

The matrix $C$ in figure 1, functions as a clause list. This list represents an ordered set of clauses that form the proof. The query clauses are clamped onto this area

and activate hard constraints that force the rest of the units of the matrix to form a valid proof (if it exists).

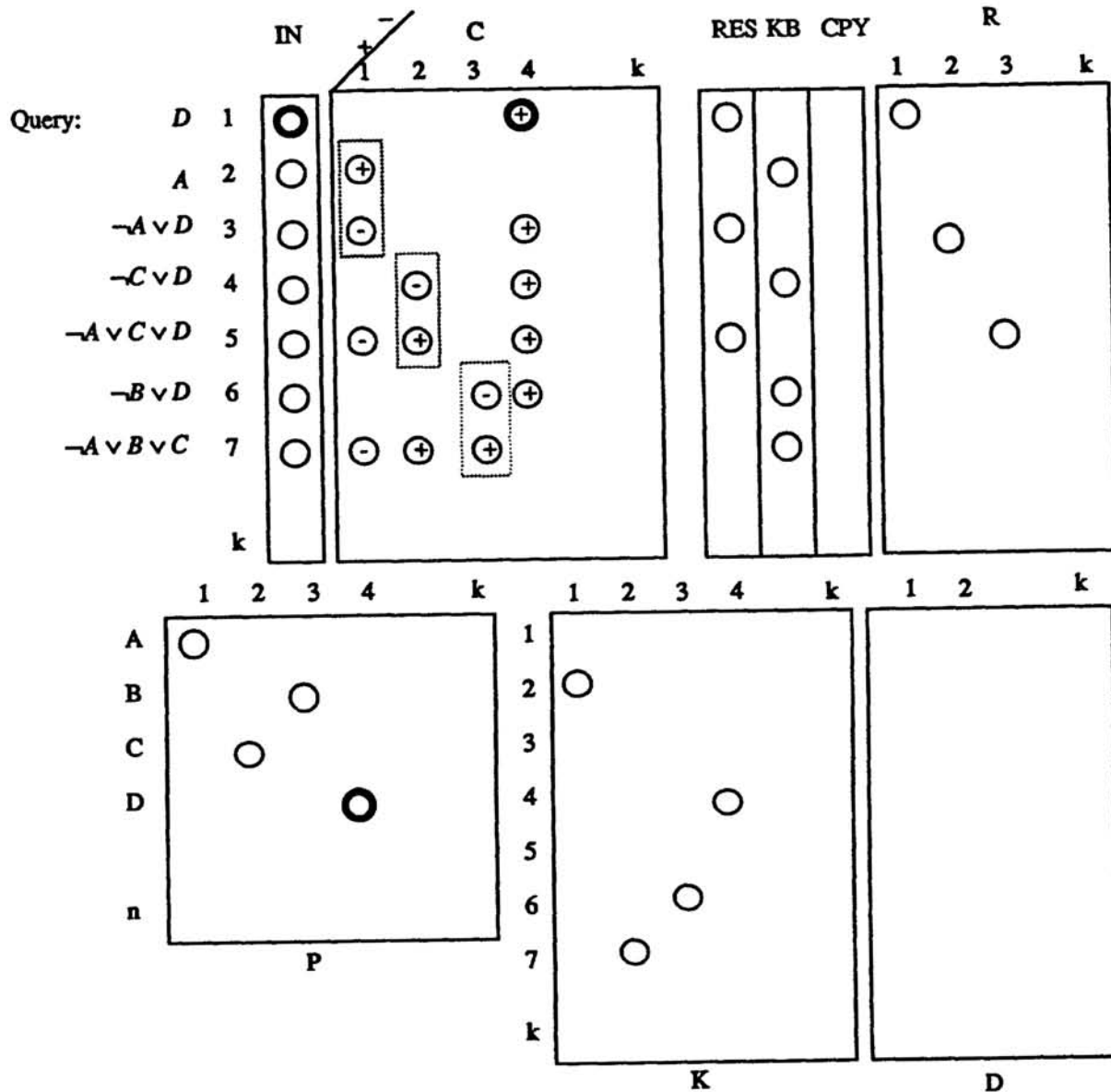

Figure 1: The proof area for a propositional case

Variable binding is performed by dynamic allocation of instances using a technique similar to [Anandan et al. 89] and [Barnden 91]. In this technique, if two symbols need to be bound together, an instance is allocated from a pool of general purpose instances, and is connected to both symbols. An instance can be connected to a literal in a clause, to a predicate type, to a constant, to a function or to a slot of another instance (for example, a constant that is bound to the first slot of a predicate).

The clauses that participate in the proof are represented using a 3-dimensional matrix ($C_{s,i,j}$) and a 2-dimensional matrix ($P_{i,j}$) as illustrated in figure 1. The rows of $C$ represent clauses of the proof, while the rows of $P$ represent atomic

propositions. The columns of both matrices represent the pool of instances used for binding propositions to clauses.

A clause is a list of negative and positive instances that represent literals. The instance thus behaves as a two-way pointer that binds composite structures like clauses with their constituents (the atomic propositions). A row $i$ in the matrix $C$ represents a clause which is composed of pairs of instances. If the unit $C_{+,i,j}$ is set, then the matrix represents a positive literal in clause $i$. If $P_{A,j}$ is also set, then $C_{+,i,j}$ represents a *positive* literal of clause $i$ that is bound to the atomic proposition $A$. Similarly $C_{-,i,j}$ represents a *negative* literal.

The first row of matrix $C$ in the figure is the query clause $D$. It contains only one positive literal that is bound to atomic proposition $D$ via instance 4. For another example consider the third row of the $C$ which represents a clause of two literals: a positive one that is bound to $D$ via instance 4, and a negative one bound to $A$ via instance 1 (it is the clause $\neg A \vee D$, generated as a result of a resolution step).

**Participation in the proof:** The vector $IN$ represents whether clauses in $C$ participate in the proof. In our example, all the clauses are in the proof; however, in the general case some of the rows of $C$ may be meaningless. When $IN_i$ is on, it means that the clause $i$ is in the proof and must be proved as well. Every clause that participates in the proof is either a result of a resolution step ($RES_i$ is set), a copy of a some clause ($CPY_i$ is set), or it is an original clause from the knowledge base ($KB_i$ is set). The second clause of $C$ in figure 1 for example is an original clause of the knowledge base. If a clause $j$ is copied, it must be in the proof itself and therefore $IN_j$ is set. Similarly, if clause $i$ is a result of a resolution step, then the two resolved clauses must also be in the proof ($IN_{i+1,j}$ and $IN_{i+2,j}$) and therefore must be themselves resolvents, copies or originals. This chain of constraints continues until all constraints are satisfied and a valid proof is generated.

**Posting a query:** The user posts a query clamping its clauses onto the first rows of $C$ and setting the appropriate $IN$ units. This indicates that the query clauses participate in the proof and should be proved by either a resolution step, a copy step or by an original clause. Figure 1 represents the complete proof for the query $D$. We start by allocating an instance (4) for $D$ in the $P$ matrix, and clamping a positive literal $D$ in the first row of $C$ ($C_{+,1,4}$); the rest of the first row's units are clamped zero. The unit $IN_1$ is biased (to have the value of one), indicating that the query is in the proof; this cause a chain of constraints to be activated that are satisfied only by a valid proof. If no proof exists, the $IN_1$ unit will become zero; i.e., the global minima is obtained by setting $IN_1$ to zero despite the bias.

**Representing resolutions steps:** The vector $RES$ is a structure of units that indicates which are the clauses in $C$ that are obtained by a resolution step. If $RES_i$ is set, then the $i$th row is obtained by resolving row $i + 1$ of $C$ with row $i + 2$. Thus, the unit $RES_1$ in figure 1 indicates that the clause $D$ of the first row of $C$ is a resolvent of the second and the third rows of $C$ representing $\neg A \vee D$ and $A$ respectfully. Two literals cancel each other if they have opposite signs and are represented by the same instance. In figure 1, literal $A$ of the third row of $C$ and literal $\neg A$ of the second row cancel each other, generating the clause of the first row.

The rows of matrix $R$ represent literals canceled by resolution steps. If row $i$ of

$C$ is the result of a resolution step, there must be one and only one instance $j$ such that both clause $i+1$ and clause $i+2$ include it with opposite signs. For example (figure 1): clause $D$ in the first row of $C$ is the result of resolving clause $A$ with clause $\neg A \vee D$ which are in the second and third rows of $C$ respectfully. Instance 1, representing atomic proposition $A$, is the one that is canceled; $R_{1,1}$ is set therefore, indicating that clause 1 is obtained by a resolution step that cancels the literals of instance 1.

**Copied and original clauses:** The matrix $D$ indicates which clauses are copied to other clauses in the proof area. Setting $D_{i,j}$ means that clause $i$ is obtained by copying (or weakening) clause $j$ into clause $i$ (the example does not use copy steps).

The matrix $K$ indicates which original knowledge-base clauses participate in the proof. The unit $K_{i,j}$ indicates that a clause $i$ in the proof area is an original clause, and the syntax of the $j$-th clause in the knowledge base must be imposed on the units of clause $i$. In figure 1 for example, clause 2 in the proof (the second row in $C$), assumes the identity of clause number 1 in the knowledge base and therefore $K_{1,2}$ is set.

## 3   Constraints

We are now ready to specify the constraints that must be satisfied by the units so that a proof is found. The constraints are specified as well formed logic formulas. For example the formula $(A \vee B) \wedge C$ imposes a constraint over the units $(A, B, C)$ such that the only possible valid assignments to those units are $(011), (101), (111)$. A general method to implement an arbitrary logical constraint on connectionist networks is shown in [Pinkas 90b]. Most of the constraints specified in this section are hard constraints; i.e., must be satisfied for a valid proof to emerge. Towards the end of this section, some soft constraints are presented.

**In-proof constraints:** If a clause participates in the proof, it must be either a result of a resolution step, a copy step or an original clause. In logic, the constraints may be expressed as: $\forall i : IN_i \rightarrow RES_i \vee CPY_i \vee KB_i$. The three units (per clause $i$) consist a winner takes all subnetwork (WTA). This means that only one of the three units is actually set. The WTA constraints may be expressed as:
$RES_i \rightarrow \neg CPY_i \wedge \neg KB_i$
$CPY_i \rightarrow \neg RES_i \wedge \neg KB_i$
$KB_i \rightarrow \neg RES_i \wedge \neg CPY_i$
The WTA property may be enforced by inhibitory connections between every pair of the three units.

**Copy constraints:** If $CPY_i$ is set then clause $i$ must be a copy of another clause $j$ in the proof. This can be expressed as $\forall i : CPY_i \rightarrow \bigvee_j (D_{i,j} \wedge IN_j)$. The rows of $D$ are WTAs allowing $i$ to be a copy of only one $j$. In addition, if clause $j$ is copied or weakened into clause $i$ then every unit set in clause $j$ must also be set in clause $i$. This may be specified as: $\forall i, j, l : D_{i,j} \rightarrow ((C_{+,i,l} \leftarrow C_{+,j,l}) \wedge (C_{-,i,l} \leftarrow C_{-,j,l}))$.

**Resolution constraints:** If a clause $i$ is a result of resolving the two clauses $i+1$ and $i+2$, then there must be one and only one instance ($j$) that is canceled (represented by $R_{i,j}$), and $C_i$ is obtained by copying both the instances of $C_{i+1}$ and $C_{i+2}$, without the instance $j$. These constraints may be expressed as:

$$\forall i : RES_i \rightarrow \bigvee_j R_{i,j} \qquad \text{at least one instance is canceled}$$
$$\forall i, j, j', j' \neq j : R_{i,j} \rightarrow \neg R_{i,j'} \qquad \text{only one instance is canceled (WT}_I$$
$$\forall i, j : R_{i,j} \rightarrow (C_{+,i+1,j} \wedge C_{-,i+2,j}) \vee (C_{-,i+1,j} \wedge C_{+,i+2,j}) \quad \text{cancel literals with opposite signs.}$$
$$\forall i : RES_i \rightarrow IN_{i+1} \wedge IN_{i+2} \qquad \text{the two resolvents are also in proof}$$
$$\forall i : RES_i \rightarrow (C_{+,i,j} \leftrightarrow (C_{+,i+1,j} \vee C_{+,i+2,j}) \wedge \neg R_{i,j} \qquad \text{copy positive literals}$$
$$\forall i : RES_i \rightarrow (C_{-,i,j} \leftrightarrow (C_{-,i+1,j} \vee C_{-,i+2,j}) \wedge \neg R_{i,j} \qquad \text{copy negative literals}$$

**Clause-instance constraints:** The sign of an instance in a clause should be unique; therefore, any instance pair in the matrix $C$ is WTA: $\forall i, j : C_{+,i,j} \rightarrow \neg C_{-,i,j}$. The columns of matrix $P$ are WTAs since an instance is allowed to represent only one atomic proposition: $\forall A, i, B \neq A : P_{A,i} \rightarrow \neg P_{B,i}$. The rows of $P$ may be also WTAs: $\forall A, i, j \neq i : P_{A,i} \rightarrow \neg P_{A,j}$ (this constraint is not imposed in the FOL case).

**Knowledge base constraints:** If a clause $i$ is an original knowledge base clause, then there must be a clause $j$ (out of the $m$ original clauses) whose syntax is forced upon the units of the $i$-th row of matrix $C$. This constraint can be expressed as: $\forall i : KB_i \rightarrow \bigvee_j^m K_{i,j}$. The rows of $K$ are WTA networks so that only one original clause is forced on the units of clause $i$: $\forall i, j, j' \neq j : K_{i,j} \rightarrow \neg K_{i,j'}$.

The only hard constraints that are left are those that force the syntax of a particular clause from the knowledge base. Assume for example that $K_{i,4}$ is set, meaning that clause $i$ in $C$ must have the syntax of the fourth clause in the knowledge base of our example ($\neg C \vee D$). Instances $j$ and $j'$ must be allocated to the atomic propositions $C$ and $D$ respectfully, and must appear also in clause $i$ as the literals $C_{-,i,j}$ and $C_{+,i,j'}$. The following constraints capture the syntax of ($\neg C \vee D$):

$$\forall i : K_{i,4} \rightarrow \bigvee_j (C_{-,i,j} \wedge P_{C,j}) \qquad \text{there exists a negative literal that is bound to } C;$$
$$\forall i : K_{i,4} \rightarrow \bigvee_j (D_{+,i,j} \wedge P_{C,j}) \qquad \text{there exists a positive literal that is bound to } D.$$

**FOL extension:**
In first-order predicate logic (FOL) instead of atomic propositions we must deal with predicates (see [Pinkas 91j] for details). As in the propositional case, a literal in a clause is represented by a positive or negative instance; however, the instance must be allocated now to a predicate name and may have slots to be filled by other instances (representing functions and constants). To accommodate such complexity a new matrix ($NEST$) is added, and the role of matrix $P$ is revised.

The matrix $P$ must accommodate now function names, predicate names and constant names instead of just atomic propositions. Each row of $P$ represents a name, and the columns represent instances that are allocated to those names. The rows of $P$ that are associated with predicates and functions may contain several different instances of the same predicate or function, thus, they are not WTA anymore. In order to represent compound terms and predicates, instances may be bound to slots of other instances. The new matrix ($NEST_{i,j,p}$) is capable of representing such bindings. If $NEST_{i,j,p}$ is set, then instance $i$ is bound to the $p$ slot of instance $j$. The columns of $NEST$ are WTA, allowing only one instance to be bound to a certain slot of another instance. When a clause $i$ is forced to have the syntax of some original clause $l$, syntactic constraints are triggered so that the literals of clause $i$ become instantiated by the relevant predicates, functions, constants and variables imposed by clause $l$.

Unification is implicitly obtained if two predicates are representing by the same instance while still satisfying all the constraints (imposed by the syntax of the two clauses). When a resolution step is needed, the network tries to allocate the same instance to the two literals that need to cancel each other. If the syntactic constraints on the literals permit such sharing of an instance, then the attempt to share the instance is successful and a unification occurs (occur check is done implicitly since the matrix $NEST$ allows the only finite trees to be represented).

**Minimizing the violation of soft constraints:** Among the valid proofs some are preferable to others. By means of soft constraints and optimization it is possible to encourage the network to search for preferred proofs. Theorem-proving thus is viewed as a constraint optimization problem. A weight may be assigned to each of the constraints [Pinkas 91c] and the network tries to minimize the weighted sum of the violated constraints, so that the set of the optimized solutions is exactly the set of the preferred proofs. For example, preference of proofs with most general unification is obtained by assignment of small penalties (negative bias) to every binding of a function to a position of another instance (in $NEST$). Using similar techniques, the network can be made to prefer shorter, more parsimonious or more reliable proofs, low-cost plans or even more specific arguments as in nonmonotonic reasoning.

## 4  Summary

Given a finite set $T$ of $m$ clauses, where $n$ is the number of different predicates, functions and constants, and given also a bound $k$ over the proof length, we can generate a network that searches for a proof with length not longer then $k$, for a clamped query $Q$. If a global minimum is found then an answer is given as to whether there exists such a proof, and the proof (with MGU's) may be extracted from the state of the visible units. Among the possible valid proofs the system prefers some "better" proofs by minimizing the violation of soft constraints. The concept of "better" proofs may apply to applications like planning (minimize the cost), abduction (parsimony) and nonmonotonic reasoning (specificity).

In the propositional case the generated network is of $O(k^2 + km + kn)$ units and $O(k^3 + km + kn)$ connections. For predicate logic there are $O(k^3 + km + kn)$ units and connections, and we need to add $O(k^i m)$ connections and hidden units, where $i$ is the complexity-level of the syntactic constraints [Pinkas 91j].

The results improve an earlier approach [Ballard 86]: There are no restrictions on the rules allowed; every proof no longer than the bound is allowed; the network is compact and the representation of bindings (unifications) is efficient; nesting of functions and multiple uses of rules are allowed; only one relaxation phase is needed; inconsistency is allowed in the knowledge base, and the query does not need to be negated and pre-wired (it can be clamped during query time).

The architecture discussed has a natural fault-tolerance capability: When a unit becomes faulty, it simply cannot assume a role in the proof, and other units are allocated instead.

**Acknowledgment:** I wish to thank Dana Ballard, Bill Ball, Rina Dechter, Peter Haddawy, Dan Kimura, Stan Kwasny, Ron Loui and Dave Touretzky for

helpful comments.

# References

[Anandan et al. 89] P. Anandan, S. Letovsky, E. Mjolsness, "Connectionist variable binding by optimization," *Proceedings of the 11th Cognitive Science Society* 1989.

[Ballard 86] D. H. Ballard "Parallel Logical Inference and Energy Minimization," *Proceedings of the 5th National Conference on Artificial Intelligence,* Philadelphia, pp. 203-208, 1986.

[Barnden 91] J.A. Barnden, "Encoding complex symbolic data structures with some unusual connectionist techniques," in J.A Barnden and J.B. Pollack, *Advances in Connectionist and Neural Computation Theory 1,* High-level connectionist models, Ablex Publishing Corporation, 1991.

[Derthick 88] M. Derthick "Mundane reasoning by parallel constraint satisfaction," PhD thesis, CMU-CS-88-182 Carnegie Mellon University, Sept. 1988

[Hinton, Sejnowski 86] G.E Hinton and T.J. Sejnowski, "Learning and re-learning in Boltzman Machines," in J. L. McClelland and D. E. Rumelhart, *Parallel Distributed Processing: Explorations in The Microstructure of Cognition I,* pp. 282 - 317, MIT Press, 1986.

[Hölldobler 90] S. Hölldobler, "CHCL, a connectionist inference system for Horn logic based on connection method and using limited resources," International Computer Science Institute TR-90-042, 1990.

[Hopfield 84b] J. J. Hopfield "Neurons with graded response have collective computational properties like those of two-state neurons," *Proceedings of the National Academy of Sciences 81,* pp. 3088-3092, 1984.

[Peterson, Hartman 89] C. Peterson, E. Hartman, "Explorations of mean field theory learning algorithm," *Neural Networks 2,* no. 6, 1989.

[Pinkas 90b] G. Pinkas, "Energy minimization and the satisfiability of propositional calculus," *Neural Computation 3,* no. 2, 1991.

[Pinkas 91c] G. Pinkas, "Propositional Non-Monotonic Reasoning and Inconsistency in Symmetric Neural Networks," *Proceedings of IJCAI,* Sydney, 1991.

[Pinkas 91j] G. Pinkas, "First-order logic proofs using connectionist constraint relaxation," technical report, Department of Computer Science, Washington University, WUCS-91-54, 1991.

[Shastri et al. 90] L. Shastri, V. Ajjanagadde, "From simple associations to systematic reasoning: A connectionist representation of rules, variables and dynamic bindings," technical report, University of Pennsylvania, Philadelphia, MS-CIS-90-05, 1990.

[Smolensky 86] P. Smolensky, "Information processing in dynamic systems: Foundations of harmony theory," in J.L.McClelland and D.E.Rumelhart, *Parallel Distributed Processing: Explorations in The Microstructure of Cognition I ,* MIT Press, 1986.
